# Discovering Hidden Variables:
# A Structure-Based Approach

**Gal Elidan    Noam Lotner    Nir Friedman**
Hebrew University
*{galel,noaml,nir}@cs.huji.ac.il*

**Daphne Koller**
Stanford University
*koller@cs.stanford.edu*

## Abstract

A serious problem in learning probabilistic models is the presence of *hidden* variables. These variables are not observed, yet interact with several of the observed variables. As such, they induce seemingly complex dependencies among the latter. In recent years, much attention has been devoted to the development of algorithms for learning parameters, and in some cases structure, in the presence of hidden variables. In this paper, we address the related problem of *detecting* hidden variables that interact with the observed variables. This problem is of interest both for improving our understanding of the domain and as a preliminary step that guides the learning procedure towards promising models. A very natural approach is to search for "structural signatures" of hidden variables — substructures in the learned network that tend to suggest the presence of a hidden variable. We make this basic idea concrete, and show how to integrate it with structure-search algorithms. We evaluate this method on several synthetic and real-life datasets, and show that it performs surprisingly well.

## 1   Introduction

In the last decade there has been a great deal of research focused on the problem of learning Bayesian networks (BNs) from data (e.g., [7]). An important issue is the existence of *hidden* variables that are never observed, yet interact with observed variables. Naively, one might think that, if a variable is never observed, we can simply ignore its existence. At a certain level, this intuition is correct. We can construct a network over the observable variables which is an *I-map* for the marginal distribution over these variables, i.e., captures all the dependencies among the observed variables. However, this approach is weak from a variety of perspectives. Consider, for example, the network in Figure 1(a). Assume that the data is generated from such a dependency model, but that the node $H$ is hidden. A minimal I-map for the marginal distribution is shown in Figure 1(b). From a pure representation perspective, this network is clearly less useful. It contains 12 edges rather than 6, and the nodes have much bigger families. Hence, as a representation of the process in the domain, it is much less meaningful. From the perspective of learning these networks from data, the marginalized network has significant disadvantages. Assuming all the variables are binary, it uses 59 parameters rather than 17, leading to substantial data fragmentation and thereby to nonrobust parameter estimates. Moreover, with limited amounts of data the induced network will usually omit several of the dependencies in the model.

When a hidden variable is known to exist, we can introduce it into the network and apply known BN learning algorithms. If the network structure is known, algorithms such as

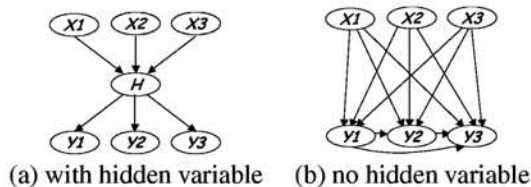

(a) with hidden variable    (b) no hidden variable

Figure 1: Hidden variable simplifies structure

*EM* [3, 9] or *gradient ascent* [2] can learn parameters. If the structure is not known, the *Structural EM (SEM)* algorithm of [4] can be used to perform structure learning with missing data. However, we cannot simply introduce a "floating" hidden variable and expect SEM to place it correctly. Hence, both of these algorithms assume that some other mechanism introduces the hidden variable in approximately the right location in the network. Somewhat surprisingly, only little work has been done on the problem of automatically detecting that a hidden variable might be present in a certain position in the network.

In this paper, we investigate what is arguably the most straightforward approach for inducing the existence of a hidden variable. This approach, briefly mentioned in [7], is roughly as follows: We begin by using standard Bayesian model selection algorithms to learn a structure over the observable variables. We then search the structure for substructures, which we call *semi-cliques*, that seem as if they might be induced by a hidden variable. We temporarily introduce the hidden variable in a way that breaks up the clique, and then continue learning based on that new structure. If the resulting structure has a better score, we keep the hidden variable. Surprisingly, this very basic technique does not seem to have been pursued. (The approach of [10] is similar on the surface, but is actually quite different; see Section 5.) We provide a concrete and efficient instantiation of this approach and show how to integrate it with existing learning algorithms such as SEM. We apply our approach to several synthetic and real datasets, and show that it often provides a good initial placement for the introduced hidden variable. We can therefore use it as a preprocessing step for SEM, substantially reducing the SEM search space.

## 2 Learning Structure of Bayesian Networks

Consider a finite set $\mathcal{X} = \{X_1, \ldots, X_n\}$ of discrete random variables where each variable $X_i$ may take on values from a finite set. A *Bayesian network* is an annotated directed acyclic graph that encodes a joint probability distribution over $\mathcal{X}$. The nodes of the graph correspond to the random variables $X_1, \ldots, X_n$. Each node is annotated with a *conditional probability distribution* that represents $P(X_i \mid \mathbf{Pa}(X_i))$, where $\mathbf{Pa}(X_i)$ denotes the parents of $X_i$ in $G$. A Bayesian network $B$ specifies a unique joint probability distribution over $\mathcal{X}$ given by: $P_B(X_1, \ldots, X_n) = \prod_{i=1}^{n} P_B(X_i \mid \mathbf{Pa}(X_i))$.

The problem of learning a Bayesian network can be stated as follows. Given a *training set* $D = \{\mathbf{x}[1], \ldots, \mathbf{x}[M]\}$ of instances of $\mathcal{X}$, find a network $B$ that *best matches* $D$. The common approach to this problem is to introduce a scoring function that evaluates each network with respect to the training data, and then to search for the best network according to this score. The scoring function most commonly used to learn Bayesian networks is the *Bayesian scoring* metric [8]. Given a scoring function, the structure learning task reduces to a problem of searching over the combinatorial space of structures for the structure that maximizes the score. The standard approach is to use a local search procedure that changes one arc at a time. Greedy hill-climbing with random restarts is typically used.

The problem of learning in the presence of partially observable data (or known hidden variables) is computationally and conceptually much harder. In the case of a fixed network structure, the *Expectation Maximization (EM)* algorithm of [3] can be used to search for a (local) maximum likelihood (or maximum a posteriori) assignment to the parameters. The *structural EM* algorithm of [4] extends this idea to the realm of structure search. Roughly speaking, the algorithm uses an E-step as part of structure search. The current *model* — structure as well as parameters — is used for computing expected sufficient statistics for

other candidate structures. The candidate structures are scored based on these expected sufficient statistics. The search algorithm then moves to a new candidate structure. We can then run EM again, for our new structure, to get the desired expected sufficient statistics.

## 3 Detecting Hidden Variables

We motivate our approach for detecting hidden variables by considering the simple example discussed in the introduction. Consider the distribution represented by the network shown in Figure 1(a), where $H$ is a hidden variable. The variable $H$ was the keystone for the conditional independence assumptions in this network. As a consequence, the marginal distribution over the remaining variables has almost no structure: each $Y_j$ depends on all the $X_i$'s, and the $Y_j$'s themselves are also fully connected. A minimal I-map for this distribution is shown in Figure 1(b). It contains 12 edges compared to the original 6. We can show that this phenomenon is a typical effect of removing a hidden variables:

**Proposition 3.1:** *Let $G$ be a network over the variables $X_1, \ldots, X_n, H$. Let $\mathcal{I}$ be the conditional independence statements — statements of the form $I(\boldsymbol{X}; \boldsymbol{Y} \mid \boldsymbol{Z})$ — that are implied by $G$ and do not involve $H$. Let $G'$ be the graph over $X_1, \ldots, X_n$ that contains an edge from $X_i$ to $X_j$ whenever $G$ contains such an edge, and in addition: $G'$ contains a clique over the children $Y_j$ of $H$, and $G'$ contains an edge from any parent $X_i$ of $H$ to any child $Y_j$ of $H$. Then $G'$ is a minimal I-map for $\mathcal{I}$.*

We want to define a procedure that will suggest candidate hidden variables by finding structures of this type in the context of a learning algorithm. We will apply our procedure to networks induced by standard structure learning algorithms [7]. Clearly, it is unreasonable to hope that there is an exact mapping between substructures that have the form described in Proposition 3.1 and hidden variables. Learned networks are rarely an exact reflection of the minimal I-map for the underlying distribution. We therefore use a somewhat more flexible definition, which allows us to detect potential hidden variables. For a node $X$ and a set of nodes $\boldsymbol{Y}$, we define $\Delta(X; \boldsymbol{Y})$ to be the set of neighbors of $X$ (parents or children) within the subset $\boldsymbol{Y}$. We define a *semi-clique* to be a set of nodes $\boldsymbol{Q}$ where each node $X \in \boldsymbol{Q}$ is linked to at least half of $\boldsymbol{Q}$: $|\Delta(X; \boldsymbol{Q})| \geq \frac{1}{2}|\boldsymbol{Q}|$ (This revised definition is the strictest criterion that still accepts a minimally (just one neighbor missing) relaxed 4-Clique.)

We propose a simple heuristic for finding semi-cliques in the graph. We first observe that each semi-clique must contain a *seed* which is easy to spot; this seed is a 3-vertex clique.

**Proposition 3.2:** *Any semi-clique of size 4 or more contains a clique of size 3.*

The first phase of the algorithm is a search for all 3-cliques in the graph. The algorithm then tries to expand each of them into a maximal semi-clique in a greedy way. More precisely, at each iteration the algorithm attempts to add a node to the "current" semi-clique. If the expanded set satisfies the semi-clique property, then it is set as the new "current" clique. These tests are repeated until no additional variable can be added to the semi-clique. The algorithm outputs the expansions found based on the different 3-clique "seeds". We note that this greedy procedure does not find all semi-cliques. The exceptions are typically two semi-cliques that are joined by a small number of edges, making a larger legal semi-clique. These cases are of less interest to us, because they are less likely to arise from the marginalization of a hidden variable.

In the second phase, we convert each of the semi-cliques to a structure *candidate* containing a new hidden node. Suppose $\boldsymbol{Q}$ is a semi-clique. Our construction introduces a new variable $H$, and replaces all of the incoming edges into variables in $\boldsymbol{Q}$ by edges from $H$. Parents of nodes in $\boldsymbol{Q}$ are then made to be parents of $H$, unless the edge results in a cycle. This process results in the removal of all intra-clique edges and makes $H$ a proxy for all "outside" influences on the nodes in the clique.

In the third phase, we evaluate each of these candidate structures in attempt to find the most useful hidden variable. There are several possible ways in which this candidate can

be utilized by the learning algorithm. We propose three approaches. The simplest assumes that the network structure, after the introduction of the hidden variable, is fixed. In other words, we assume that the "true" structure of the network is indeed the result of applying our transformation to the input network (which was produced by the first stage of learning). We can then simply fit the parameters using EM, and score the resulting network.

We can improve this idea substantially by noting that our simple transformation of the semi-clique does not typically recover the true underlying structure of the original model. In our construction, we chose to make the hidden variable $H$ the parent of all the nodes in the semi-clique, and eliminate all other incoming edges to variables in the clique. Clearly, this construction is very limited. There might well be cases where some of the edges in the clique are warranted even in the presence of the hidden variable. It might also be the case that some of the edges from $H$ to the semi-clique variables should be reversed. Finally, it is plausible that some nodes were included in the semi-clique accidentally, and should not be directly correlated with $H$. We could therefore allow the learning algorithm — the SEM algorithm of [4] — to adapt the structure after the hidden variable is introduced. One approach is to use SEM to fine-tune our model for the part of the network we just changed: the variables in the semi-clique and the new hidden variable. Therefore, in the second approach we fix the remaining structure, and consider only adaptations of the edges within this set of variables. This restriction substantially reduces the search space for the SEM algorithm. The third approach allows full structural adaptation over the entire network. This offers the SEM algorithm greater flexibility, but is computationally more expensive.

To summarize our approach: In the first phase we analyze the network learned using conventional structure search to find semi-cliques that indicate potential locations of hidden variables. In the second phase we convert these semi-cliques into structure candidates (each containing a new hidden variable). Finally, in the third phase we evaluate each of these structures (possibly using them as a seed for further search) and return the best scoring network we find.

The main assumption of our approach is that we can find "structural signatures" of hidden variables via semi-cliques. As we discussed above, it is unrealistic to expect the learned network $G$ to have exactly the structure described in Proposition 3.1. On the one hand, learned networks often have spurious edges resulting from statistical noise, which might cause fragments of the network to resemble these structures even if no hidden variable is involved. On the other hand, there might be edges that are missing or reversed. Spurious edges are less problematic. At worst, they will lead us to propose a spurious hidden variable which will be eliminated by the subsequent evaluation step. Our definition of semi-clique, with its more flexible structure, partially deals with the problem of missing edges. However, if our data is very sparse, so that standard learning algorithms will be very reluctant to produce clusters with many edges, the approach we propose will not work.

## 4   Experimental Results

Our aim is to evaluate the success of our procedure in detecting hidden variables. To do so, we evaluated our procedure on both synthetic and real-life data sets. The synthetic data sets were sampled from Bayesian networks that appear in the literature. We then created a training set in which we "hid" one variable. We chose to hide variables that are "central" in the network (i.e., variables that are the parents of several children). The synthetic data sets allow for a controlled evaluation, and for generating training and testing data sets of any desired size. However, the data is generated from a distribution that indeed has only a single hidden variable. A more realistic benchmark is real data, that may contain many confounding influences. In this case, of course, we do not have a generating model to compare against.

**Insurance**: A 27-node network developed to evaluate driver's insurance applications [2]. We hid the variables *Accident*, *Age*, *MakeModel*, and *VehicleYear* ($A$, $G$, $M$, $V$ in Figure 2). **Alarm**: A 37-node network [1] developed to monitor ICU patients. We hid the variables *HR*, *Intubation*, *LVFailure*, and *VentLung* ($H$, $I$, $L$, $V$ in Figure 2). **Stock Data**:

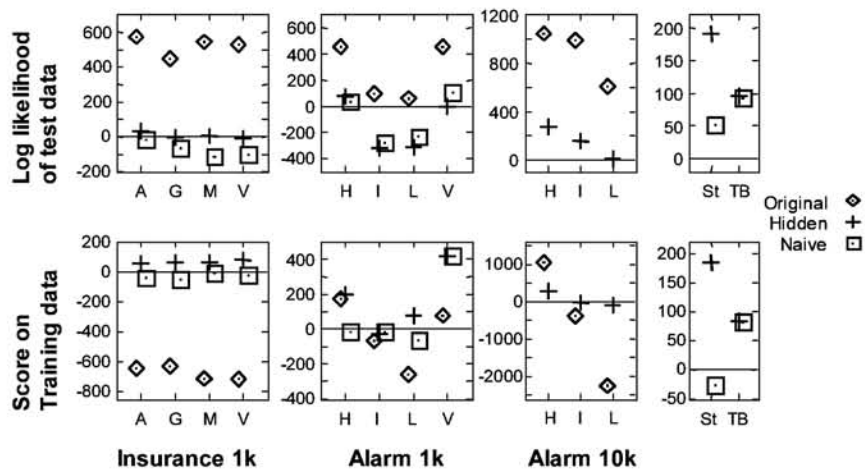

Figure 2: Comparison of the different approaches. Each point in the graph corresponds to a network learned by one of the methods. The graphs on the bottom row show the log of the Bayesian score. The graphs on the top row show log-likelihood of an independent test set. In all graphs, the scale is normalized to the performance of the **No-hidden** network, shown by the dashed line at "0".

A real-life dataset that traces the daily change of 20 major US technology stocks for several years (1516 trading days). These values were discretized to three categories: "up", "no change", and "down". **TB**: A real-life dataset that records information about 2302 tuberculosis patients in the San Francisco county (courtesy of Dr. Peter Small, Stanford Medical Center). The data set contains demographic information such as gender, age, ethnic group, and medical information such as HIV status, TB infection type, and other test results.

In each data set, we applied our procedure as follows. First, we used a standard model selection procedure to learn a network from the training data (without any hidden variables). In our implementation, we used standard greedy hill-climbing search that stops when it reaches a plateau it cannot escape. We supplied the learned network as input to the clique-detecting algorithm which returned a set of candidate hidden variables. We then used each candidate as the starting point for a new learning phase. The **Hidden** procedure returns the highest-scoring network that results from evaluating the different putative hidden variables.

To gauge the quality of this learning procedure, we compared it to two "strawmen" approaches. The **Naive** strawman [4] initializes the learning with a network that has a single hidden variable as parent of all the observed variables. It then applies SEM to get an improved network. This process is repeated several times, where each time a random perturbation (e.g., edge addition) is applied to help SEM to escape local maxima. The **Original** strawman, which applied only in synthetic data set, is to use the true generating network on the data set. That is, we take the original network (that contains the variable we hid) and use standard parametric EM to learn parameters for it. This strawman corresponds to cases where the learner has additional prior knowledge about domain structure.

We quantitatively evaluated each of these networks in two ways. First, we computed the Bayesian score of each network on the training data. Second, we computed the logarithmic loss of predictions made by these networks on independent test data. The results are shwon in Figure 2. In this evaluation, we used the performance of **No-Hidden** as the baseline for comparing the other methods. Thus, a positive score of say 100 in Figure 2 indicates a score which is larger by 100 than the score of **No-Hidden**. Since scores are the logarithm of the Bayesian posterior probability of structures (up to a constant), this implies that such a structure is $2^{100}$ times more probable than the structure found by **No-Hidden**.

We can see that, in most cases, the network learned by **Hidden** outperforms the network learned by **No-hidden**. In the artificial data sets, **Original** significantly outperforms our algorithm on test data. This is no surprise: **Original** has complete knowledge of the structure which generated the test data. Our algorithm can only evaluate networks according to their score; indeed, the scores of the networks found by **Hidden** are better than those of **Original** in 12 out of 13 cases tested. Thus, we see that the "correct" structure does not usually have the highest Bayesian score. Our approach usually outperforms the network learned by **Naive**. This improvement is particularly significant in the real-life datasets.

As discussed in Section 3, there are three ways that a learning algorithm can utilize the original structure proposed by our algorithm. As our goal was to find the best model for the domain, we ran all three of them in each case, and chose the best resulting network. In all of our experiments, the variant that fixed the candidate structure and learned parameters for it resulted in scores that were significantly worse than the networks found by the variants that employed structure search. The networks trained by this variant also performed much worse on test data. This highlights the importance of structure search in evaluating a potential hidden variable. The initial structure candidate is often too simplified; on the one hand, it forces too many independencies among the variables in the semi-clique, and on the other, it can add too many parents to the new hidden variable.

The comparison between the two variants that use search is more complex. In many cases, the variant that gives the SEM complete flexibility in adapting the network structure did not find a better scoring network than the variant that only searches for edges in the area of the new variable. In the cases it did lead to improvement, the difference in score was not significantly larger. Since the variant that restricts SEM is computationally cheaper (often by an order of magnitude), we believe that it provides a good tradeoff between model quality and computational cost.

The structures found by our procedure are quite appealing. For example, in the stock market data, our procedure constructs a hidden variable that is the parent of several stocks: Microsoft, Intel, Dell, CISCO, and Yahoo. A plausible interpretation of this variable is "strong" market vs. "stationary" market. When the hidden variable has the "strong" value, all the stocks have higher probability for going up. When the hidden variable has the "stationary" probability, these stocks have much higher probability of being in the "no change" value. We do note that in the learned networks there were still many edges between the individual stocks. Thus, the hidden variable serves as a general market trend, while the additional edges make better description of the correlations between individual stocks. The model we learned for the TB patient dataset was also interesting. One value of the hidden variable captures two highly dominant segments of the population: older, HIV-negative, foreign-born Asians, and younger, HIV-positive, US-born blacks. The hidden variable's children distinguished between the two aggregated subpopulations using the *HIV-result* variable, which was also a parent of most of them. We believe that, had we allowed the hidden variable to have three values, it would have separated these populations.

## 5  Discussion and Future Work

In this paper, we propose a simple and intuitive algorithm for finding plausible locations for hidden variables in BN learning. It attempts to detect structural signatures of a hidden variable in the network learned by standard structure search. We presented experiments showing that our approach is reasonably successful at producing better models. To our knowledge, this paper is also the first to provide systematic empirical tests of any approach to the task of discovering hidden variables.

The problem of detecting hidden variables has received surprisingly little attention. Spirtes *et al.* [11] suggest an approach that detects patterns of conditional independencies that can only be generated in the presence of hidden variables. This approach suffers from two limitations. First, it is sensitive to failure in few of the multiple independence tests it uses. Second, it only detects hidden variables that are *forced* by the qualitative independence constraints. It cannot detect situations where the hidden variable provides a more succinct

model of a distribution that can be described by a network without a hidden variable (as in the simple example of Figure 1).

Martin and VanLehn [10] propose an alternative approach that appears, on the surface, to be similar to ours. They start by checking correlations between all pairs of variables. This results in a "dependency" graph in which there is an edge from $X$ to $Y$ if their correlation is above a predetermined threshold. Then they construct a two-layered network that contains independent hidden variables in the top level, and observables in the bottom layer, such that every dependency between two observed variables is "explained" by at least one common hidden parent. This approach suffers from three important drawbacks. First, it does not eliminate from consideration correlations that can be explained by direct edges among the observables. Thus, it forms clusters even in cases where the dependencies can be fully explained by a standard Bayesian network structure. Moreover, since it only examines pairwise dependencies, it cannot detect conditional independencies, such as $X \to Y \to Z$, from the data. (In this case, it would learn a hidden variable that is the parent of all three variables.) Finally, this approach learns a restricted form of networks that requires many hidden variables to represent dependencies among variables. Thus, it has limited utility in distinguishing "true" hidden variables from artifacts of the representation.

We plan to test further enhancements to the algorithm in several directions. First, other possibilities for structural signatures (for example the structure resulting from a many parent – many children configuration) may expand the range of variables we can discover. Second, our clique-discovering procedure is based solely on the structure of the network learned. Additional information, such as the confidence of learned edges [6, 5], might help the procedure avoid spurious signatures. Third, we plan to experiment with multi-valued hidden variables and better heuristics for selecting candidates out of the different proposed networks. Finally, we are considering approaches for dealing with sparse data, when the structural signatures do not manifest. Information-theoretic measures might provide a more statistical signature for the presence of a hidden variable.

## Acknowledgements

This work was supported in part by ISF grant 244/99, Israeli Ministry of Science grant 2008-1-99. Nir Friedman was supported by Alon fellowship, and by the generosity of the Sacher foundation.

## References

[1] I. Beinlich, G. Suermondt, R. Chavez, and G. Cooper. The ALARM monitoring system. In *Proc. 2'nd European Conf. on AI and Medicine.* , 1989.

[2] J. Binder, D. Koller, S. Russell, and K. Kanazawa. Adaptive probabilistic networks with hidden variables. *Machine Learning*, 29:213–244, 1997.

[3] A. P. Dempster, N. M. Laird, and D. B. Rubin. Maximum likelihood from incomplete data via the EM algorithm. *J. Royal Stat. Soc.*, B 39:1–39, 1977.

[4] N. Friedman. The Bayesian structural EM algorithm. In *UAI*, 1998.

[5] N. Friedman and D. Koller. Being Bayesian about Network Structure. In *UAI*, 2000.

[6] N. Friedman, M. Goldszmidt, and A. Wyner. Data analysis with Bayesian networks: A bootstrap approach. In *UAI*, 1999.

[7] D. Heckerman. A tutorial on learning with Bayesian networks. In *Learning in Graphical Models*. 1998.

[8] D. Heckerman, D. Geiger, and D. M. Chickering. Learning Bayesian networks: The combination of knowledge and statistical data. *Machine Learning*, 20:197–243, 1995.

[9] S. L. Lauritzen. The EM algorithm for graphical association models with missing data. *Comp. Stat.and Data Ana.*, 19:191–201, 1995.

[10] J. Martin and K. VanLehn. Discrete factor analysis: Learning hidden variables in Bayesian networks. Technical report, Department of Computer Science, University of Pittsburgh, 1995.

[11] P. Spirtes, C. Glymour, and R. Scheines. *Causation, Prediction and Search*. Springer-Verlag, 1993.
